# A Dynamical Model of Context Dependencies for the Vestibulo-Ocular Reflex

Olivier J.M.D. Coenen*          Terrence J. Sejnowski[†]

Computational Neurobiology Laboratory
Howard Hughes Medical Institute
The Salk Institute for Biological Studies
10010 North Torrey Pines Road
La Jolla, CA 92037, U.S.A.

Departments of [†]Biology and *[†]Physics
University of California, San Diego
La Jolla, CA 92093, U.S.A
{olivier,terry}@salk.edu

## Abstract

The vestibulo-ocular reflex (VOR) stabilizes images on the retina during rapid head motions. The gain of the VOR (the ratio of eye to head rotation velocity) is typically around -1 when the eyes are focused on a distant target. However, to stabilize images accurately, the VOR gain must vary with context (eye position, eye vergence and head translation). We first describe a kinematic model of the VOR which relies solely on sensory information available from the semicircular canals (head rotation), the otoliths (head translation), and neural correlates of eye position and vergence angle. We then propose a dynamical model and compare it to the eye velocity responses measured in monkeys. The dynamical model reproduces the observed amplitude and time course of the modulation of the VOR and suggests one way to combine the required neural signals within the cerebellum and the brain stem. It also makes predictions for the responses of neurons to multiple inputs (head rotation and translation, eye position, etc.) in the oculomotor system.

## 1 Introduction

The VOR stabilizes images on the retina during rapid head motions: Rotations and translations of the head in three dimensions must be compensated by appropriate rotations of the eye. Because the head's rotation axis is not the same as the eye's rotation axis, the calculations for proper image stabilization of an object must take into account diverse variables such as object distance from each eye,

gaze direction, and head translation (Viire et al., 1986). The stabilization is achieved by integrating information from different sources: head rotations from the semicircular canals of the inner ear, head translations from the otolith organs, eye positions, viewing distance, as well as other context information, such as posture (head tilts) or activity (walking, running) (Snyder and King, 1992; Shelhamer et al.,1992; Grossman et al., 1989). In this paper we concentrate on the context modulation of the VOR which can be described by the kinematics of the reflex, i.e. eye position, eye vergence and head translation.

## 2   The Vestibulo-Ocular Reflex: Kinematic Model

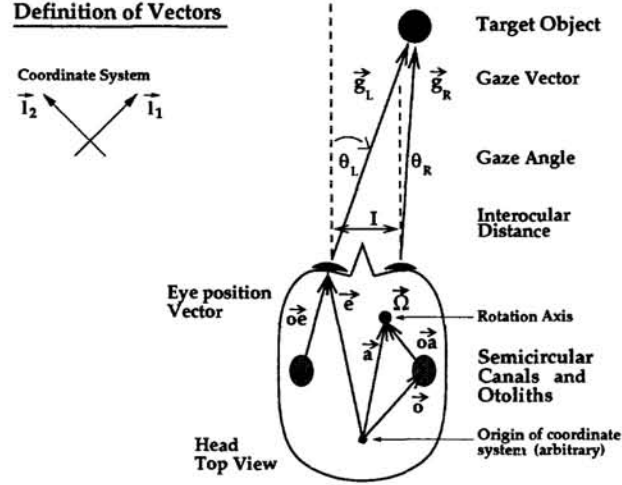

Figure 1: Diagram showing the definition of the vectors used in the equation of the kinematic model of the vestibulo-ocular reflex.

The ideal VOR response is a compensatory eye movement which keeps the image fixed on the retina for any head rotations and translations. We therefore derived an equation for the eye rotation velocity by requiring that a target remains stationary on the retina. The velocity of the resulting compensatory eye rotation can be written as (see fig. 1):

$$\vec{\omega} = -\vec{\Omega}_c + \frac{\hat{g}}{|g|} \times \left[ \vec{oe}_j \times \vec{\Omega}_c - \vec{T}_{o_j} \right] \tag{1}$$

where $\vec{\Omega}_c$ is the head rotation velocity sensed by the semicircular canals, $\vec{T}_{o_j}$ is the head translation velocity sensed by the otoliths, $\vec{oe}_j \equiv (\vec{e} - \vec{o}_j)$, $\vec{e}$ is a constant vector specifying the location of an eye in the head, $\vec{o}_j$ is the position of either the left or right otolith, $\hat{g}$ and $|g|$ are the unit vector and amplitude of the gaze vector: $\hat{g}$ gives the eye position (orientation of the eye relative to the head), and $|g|$ gives the distance from the eye to the object, and the symbol $\times$ indicates the cross-product between two vectors. $\vec{\omega}$ and $\vec{\Omega}_c$ are rotation vectors which describe the instantaneous angular velocity of the eye and head, respectively. A rotation vector lies along the instantaneous axis of rotation; its magnitude indicates the speed of rotation around the axis, and its direction is given by the right-hand screw rule. A motion of the head combining rotation ($\vec{\Omega}$) and translation ($\vec{T}$) is sensed as the combination of a rotation velocity $\vec{\Omega}_c$ measured by the semicircular canals and a translation velocity $\vec{T}_o$ sensed by the otoliths. The rotation vectors are equal ($\vec{\Omega} = \vec{\Omega}_c$), and the translation velocity vector as measured by the otoliths is given by: $\vec{T}_{o_j} = \vec{oa}_j \times \vec{\Omega} + \vec{T}$, where $\vec{oa}_j \equiv (\vec{a} - \vec{o}_j)$, and $\vec{a}$ is the position vector of the axis of rotation.

The special case where the gaze is horizontal and the rotation vector is vertical (horizontal head rotation) has been studied extensively in the literature. We used this special case in the simulations. In that case $\vec{w}$ may be simplify by writing its equation with dot products. Since $\hat{g}$ and $\vec{\Omega}_c$ are then perpendicular ($\hat{g} \cdot \vec{\Omega}_c = 0$), the first term of the following expression in brackets is zero:

$$\vec{\omega} = -\vec{\Omega}_c + \frac{1}{|g|}\left[\vec{oe}(\hat{g} \cdot \vec{\Omega}_c) - \vec{\Omega}_c(\hat{g} \cdot \vec{oe}) - \hat{g} \times \vec{T}_o\right] \tag{2}$$

The semicircular canals decompose and report acceleration and velocity of head rotation $\vec{\Omega}$ by its components along the three canals on each side of the head $\vec{\Omega}_c$ : horizontal, anterior and posterior. The two otolith organs on each side report the dynamical inertial forces generated during linear motion (translation) in two perpendicular plane, one vertical and the other horizontal relative to the head. Here we assume that a translation velocity signal ($\vec{T}_o$) derived from or reported by the otolith afferents is available. The otoliths encode as well the head orientation relative to the gravity vector force, but was not included in this study.

To complete the correspondence between the equation and a neural correlate, we need to determine a physiological source for $\hat{g}$ and $\frac{1}{|g|}$. The eye position $\hat{g}$ is assumed to be given by the output of the velocity-to-position transformation or so-called "neural integrator" which provides eye position information and which is necessary for the activation of the motoneuron to sustain the eye in a fixed position. The integrator for horizontal eye position appears to be located in the nucleus prepositus hypoglossi in the pons, and the vertical integrator in the midbrain interstitial nucleus of Cajal. (Crawford, Cadera and Vilis, 1991; Cannon and Robinson, 1987). We assume that the eye position is given as the coordinates of the unit vector $\hat{g}$ along the $\vec{l}_1$ and $\vec{l}_2$ of fig. 1. The eye position depends on the eye velocity according to $\frac{d\hat{g}}{dt} = \hat{g} \times \vec{w}$. For the special case $\vec{w}(t) = w(t)\hat{z}$, i.e. for horizontal head rotation, the eye position coordinates are given by:

$$\begin{aligned}
\hat{g}_1(t) &= \hat{g}_1(0) + \int_0^t \hat{g}_2(\tau)w(\tau)\,d\tau \\
\hat{g}_2(t) &= \hat{g}_2(0) - \int_0^t \hat{g}_1(\tau)w(\tau)\,d\tau
\end{aligned} \tag{3}$$

This is a set of two negatively coupled integrators. The "neural integrator" therefore does not integrate the eye velocity directly but a product of eye position and eye velocity. The distance from eye to target $\frac{1}{|g|}$ can be written using the gaze angles in the horizontal plane of the head:

$$\text{Right eye:} \quad \frac{1}{|g_R|} = \frac{\sin(\theta_R - \theta_L)}{I\cos(\theta_L)} = \frac{1}{I}\sec(\theta_L)\sin(\theta_R - \theta_L) \tag{4}$$

$$\text{Left eye:} \quad \frac{1}{|g_L|} = \frac{\sin(\theta_R - \theta_L)}{I\cos(\theta_R)} = \frac{1}{I}\sec(\theta_R)\sin(\theta_R - \theta_L) \tag{5}$$

where $(\theta_R - \theta_L)$ is the vergence angle, and $I$ is the interocular distance; the angles are measured from a straight ahead gaze, and take on negative values when the eyes are turned towards the right. Within the oculomotor system, the vergence angle and speed are encoded by the mesencephalic reticular formation neurons (Judge and Cumming, 1986; Mays, 1984). The nucleus reticularis tegmenti pontis with reciprocal connections to the flocculus, oculomotor vermis, paravermis of the cerebellum also contains neurons which activity varies linearly with vergence angle (Gamlin and Clarke, 1995).

We conclude that it is possible to perform the computations needed to obtain an ideal VOR with signals known to be available physiologically.

Dynamical Model Overview

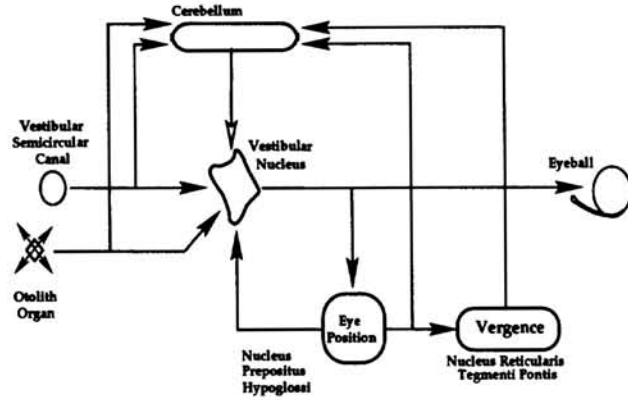

Figure 2: Anatomical connections considered in the dynamical model. Only the left side is shown, the right side is identical and connected to the left side only for the calculation of vergence angle. The nucleus prepositus hypoglossi and the nucleus reticularis tegmenti pontis are meant to be representative of a class of nuclei in the brain stem carrying eye position or vergence signal. All connections are known to exist except the connection between the prepositus nucleus to the reticularis nucleus which has not been verified. Details of the cerebellum are in fig. 3 and of the vestibular nucleus in fig. 4.

## 3   Dynamical Model

Snyder & King (1992) studied the effect of viewing distance and location of the axis of rotation on the VOR in monkeys; their main results are reproduced in fig. 5. In an attempt to reproduce their data and to understand how the signals that we have described in section 2 may be combined in time, we constructed a dynamical model based on the kinematic model. Its basic anatomical structure is shown in fig. 2. Details of the model are shown in fig. 3, and fig. 4 where all constants are written using a millisecond time scale. The results are presented in fig. 5. The dynamical variables represent the change of average firing rate from resting level of activity. The firing rate of the afferents has a tonic component proportional to the velocity and a phasic component proportional to the acceleration of movement. Physiologically, the afferents have a wide range of phasic and tonic amplitudes. This is reflected by a wide selection of parameters in the numerators in the boxes of fig. 3 and fig. 4. The Laplace transform of the integration operator in equation (3) of the eye position coordinates is $\frac{1}{s}$. Following Robinson (1981), we modeled the neural integrator with a gain and a time constant of 20 seconds. We therefore replaced the pure integrator $\frac{1}{s}$ with $\frac{20000}{20000s+1}$ in the calculations of eye position. The term $\frac{1}{g}$ in fig. 3 is calculated by using equations (4) and (5), and by using the integrator $\frac{20000}{20000s+1}$ on the eye velocity motor command to find the angles $\theta_L$ and $\theta_R$.

The dynamical model is based on the assumption that the cerebellum is required for context modulation, and that because of its architecture, the cerebellum is more likely to implement complex functions of multiple signals than other relevant nuclei. The major contributions of vergence and eye position modulation on the VOR are therefore mediated by the cerebellum. Smaller and more transient contributions from eye position are assumed to be mediated through the vestibular nucleus as shown in fig. 4. The motivation for combining eye position as in fig. 4 are, first, the evidence for eye response oscillations; second, the theoretical consideration that linear movement information ($\vec{T}_o$) is useless without eye position information for proper VOR.

The parameters in the dynamical model were adjusted by hand after observing the behavior of the different components of the model and noting how these combine to produce the oscillations observed

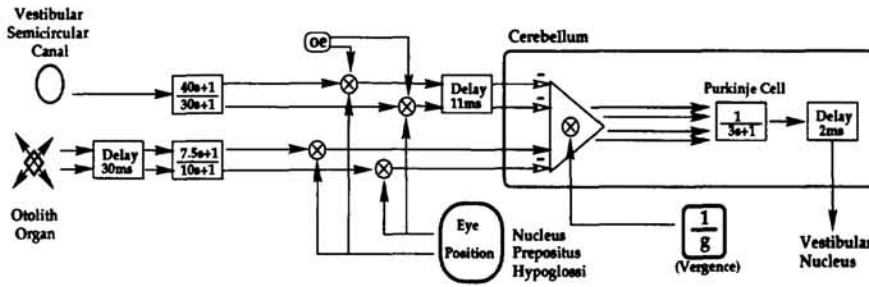

Figure 3: Contribution of the cerebellum to the dynamical model. Filtered velocity inputs from the canals and otoliths are combined with eye position according to equation (2). These calculations could be performed either outside the cerebellum in one or multiple brain stem nuclei (as shown) or possibly inside the cerebellum. The only output is to the vestibular nucleus. The Laplace notation is used in each boxes to represent a leaky integrator with a time constant, input derivative and input gain. The term oe are the coordinates of the vector $\vec{oe}$ shown in fig. 1. The $\times$ indicates a multiplication. The term $\frac{1}{g}$ multiplies each inputs individually. The open arrows indicate inhibitory (negative) connections.

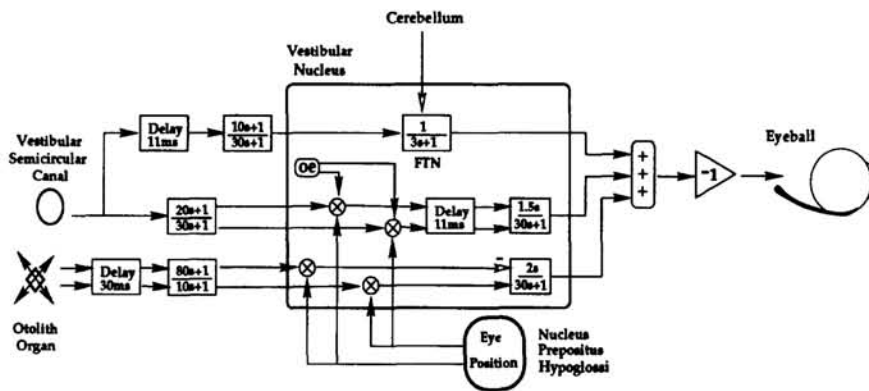

Figure 4: Contribution of the vestibular nucleus to the dynamical model. Three pathways in the vestibular nucleus process the canal and otolith inputs to drive the eye. The first pathway is modulated by the output of the cerebellum through a FTN (Flocculus Target Neuron). The second and third pathways report transient information from the inputs which are combined with eye position in a manner identical to fig. 3. The location of these calculations is hypothetical.

in the data. Even though the number of parameters in the model is not small, it was not possible to fit any single response in fig. 5 without affecting most of the other eye responses. This puts severe limits on the set of parameters allowed in the model.

The dynamical model suggests that the oscillations present in the data reflect: 1) important acceleration components in the neural signals, both rotational and linear, 2) different time delays between the canal and otolith signal processing, and 3) antagonistic or synergistic action of the canal and otolith signals with different axes of rotation, as described by the two terms in the bracket of equation (2).

## 4   Discussion

By fitting the dynamical model to the data, we tested the hypothesis that the VOR has a response close to ideal taking into account the time constraints imposed by the sensory inputs and the neural networks performing the computations. The vector computations that we used in the model may not

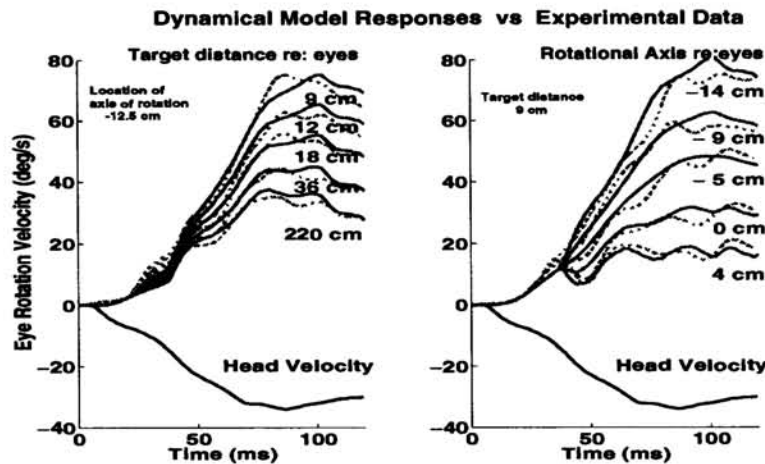

Figure 5: Comparison between the dynamical model and monkey data. The dotted lines show the effect of viewing distance and location of the axis of rotation on the VOR as recorded by Snyder & King (1992) from monkeys in the dark. The average eye velocity response (of left and right eye) to a sudden change in head velocity is shown for different target distances (left) and rotational axes (right). On the left, the location of the axis of rotation was in the midsagittal plane 12.5 cm behind the eyes (-12.5 cm), and the target distance was varied between 220 cm and 9 cm. On the right, the target distance was kept constant at 9 cm in front of the eye, and the location of the axis of rotation was varied from 14 cm behind to 4 cm in front of the eyes (-14 cm to 4 cm) in the midsagittal plane. The solid lines show the model responses. The model replicates many characteristics of the data. On the left the model captures the eye velocity fluctuations between 20-50 ms, followed by a decrease and an increase which are both modulated with target distance (50-80 ms). The later phase of the response (80-100 ms) is almost exact for 220 cm, and one peak is seen at the appropriate location for the other distances. On the right the closest fits were obtained for the 4 cm and 0 cm locations. The mean values are in good agreement and the waveforms are close, but could be shifted in time for the other locations of the axis of rotations. Finally, the latest peak (~ 100 ms) in the data appears in the model for -14 cm and 9 cm location.

be the representation used in the oculomotor system. Mathematically, the vector representation is only one way to describe the computations involved. Other representations exist such as the quaternion representation which has been studied in the context of the saccadic system (Tweed and Vilis, 1987; see also Handzel and Flash, 1996 for a very general representation). Detailed comparisons between the model and recordings from neurons will be require to settle this issue.

Direct comparison between Purkinje cell recordings (L.H. Snyder & W.M. King, unpublished data) and predictions of the model could be used to determine more precisely the different inputs to some Purkinje cells. The model can therefore be an important tool to gain insights difficult to obtain directly with experiments.

The question of how the central nervous system learns the transformations that we described still remains. The cerebellum may be one site of learning for these transformations, and its output may modulate the VOR in real time depending on the context. This view is compatible with the results of Angelaki and Hess (1995) which indicate that the cerebellum is required to correctly perform an otolith transformation. It is also consistent with adaptation results in the VOR. To test this hypothesis, we have been working on a model of the cerebellum which learns to anticipate sensory inputs and feedbacks, and use these signals to modulate the VOR. The learning in the cerebellum and vestibular nuclei is mediated by the climbing fibers which report a reinforcement signal of the prediction error (Coenen and Sejnowski, in preparation).

## 5  Conclusion

Most research on the VOR has assumed forward gaze focussed at infinity. The kinematics of off-center gaze and fixation at finite distance necessitates nonlinear corrections that require the integration of a variety of sensory inputs. The dynamical model studied here is a working hypothesis for how these corrections could be computed and is generally consistent with what is known about the cerebellum and brain stem nuclei. We are, however, far from knowing the mechanisms underlying these computations, or how they are learned through experience.

## 6  Acknowledgments

The first author was supported by a McDonnell-Pew Graduate Fellowship during this research. We would like to thank Paul Viola for helpful discussions.

### References

Angelaki, D. E. and Hess, B. J. (1995). Inertial representation of angular motion in the vestibular system of rhesus monkeyus. II. Otolith-controlled transformation that depends on an intact cerebellar nodulus. *Journal of Neurophysiology*, 73(5):1729–1751.

Cannon, S. C. and Robinson, D. A. (1987). Loss of the neural integrator of the oculomotor system from brain stem lesions in monkey. *Journal of Neurophysiology*, 57(5):1383–1409.

Crawford, J. D., Cadera, W., and Vilis, T. (1991). Generation of torsional and vertical eye position signals by the interstitial nucleus of Cajal. *Science*, 252:1551–1553.

Gamlin, P. D. R. and Clarke, R. J. (1995). Single-unit activity in the primate nucleus reticularis tegmenti pontis related to vergence and ocular accomodation. *Journal of Neurophysiology*, 73(5):2115–2119.

Grossman, G. E., Leigh, R. J., Bruce, E. N., Huebner, W. P., and Lanska, D. J. (1989). Performance of the human vestibuloocular reflex during locomotion. *Journal of Neurophysiology*, 62(1):264–272.

Handzel, A. A. and Flash, T. (1996). The geometry of eye rotations and listing's law. In Touretzky, D., Mozer, M., and Hasselmo, M., editors, *Advances in Neural Information Processing Systems 8*, Cambridge, MA. MIT Press.

Judge, S. J. and Cumming, B. G. (1986). Neurons in the monkey midbrain with activity related to vergence eye movement and accomodation. *Journal of Neurophysiology*, 55:915–930.

Mays, L. E. (1984). Neural control of vergence eye movements: Convergence and divergence neurons in midbrain. *Journal of Neurophysiology*, 51:1091–1108.

Robinson, D. A. (1981). The use of control systems analysis in the neurophysiology of eye movements. *Ann. Rev. Neurosci.*, 4:463–503.

Shelhamer, M., Robinson, D. A., and Tan, H. S. (1992). Context-specific adaptation of the gain of the vestibulo-ocular reflex in humans. *Journal of Vestibular Research*, 2:89–96.

Snyder, L. H. and King, W. M. (1992). Effect of viewing distance and location of the axis of head rotation on the monkey's vestibuloocular reflex I. eye movement response. *Journal of Neurophysiology*, 67(4):861–874.

Tweed, D. and Vilis, T. (1987). Implications of rotational kinematics for the oculomotor system in three dimensions. *Journal of Neurophysiology*, 58(4):832–849.

Viire, E., Tweed, D., Milner, K., and Vilis, T. (1986). A reexamination of the gain of the vestibuloocular reflex. *Journal of Neurophysiology*, 56(2):439–450.
